# Spike-Based Compared to Rate-Based Hebbian Learning

**Richard Kempter\***
Institut für Theoretische Physik
Technische Universität München
D-85747 Garching, Germany

**Wulfram Gerstner**
Swiss Federal Institute of Technology
Center of Neuromimetic Systems, EPFL-DI
CH-1015 Lausanne, Switzerland

**J. Leo van Hemmen**
Institut für Theoretische Physik
Technische Universität München
D-85747 Garching, Germany

## Abstract

A correlation-based learning rule at the spike level is formulated, mathematically analyzed, and compared to learning in a firing-rate description. A differential equation for the learning dynamics is derived under the assumption that the time scales of learning and spiking can be separated. For a linear Poissonian neuron model which receives time-dependent stochastic input we show that spike correlations on a millisecond time scale play indeed a role. Correlations between input and output *spikes* tend to stabilize structure formation, provided that the form of the *learning window* is in accordance with Hebb's principle. Conditions for an intrinsic normalization of the average synaptic weight are discussed.

## 1 Introduction

Most learning rules are formulated in terms of mean firing rates, viz., a continuous variable reflecting the mean activity of a neuron. For example, a 'Hebbian' (Hebb 1949) learning rule which is driven by the correlations between presynaptic and postsynaptic rates may be used to generate neuronal receptive fields (e.g., Linsker 1986, MacKay and Miller 1990, Wimbauer et al. 1997) with properties similar to those of real neurons. A rate-based description, however, neglects effects which are due to the pulse structure of neuronal signals. During recent years experimental and

theoretical evidence has accumulated which suggests that temporal coincidences between spikes on a millisecond or even sub-millisecond scale play an important role in neuronal information processing (e.g., Bialek et al. 1991, Carr 1993, Abeles 1994, Gerstner et al. 1996). Moreover, changes of synaptic efficacy depend on the precise timing of postsynaptic action potentials and presynaptic input spikes (Markram et al. 1997, Zhang et al. 1998). A synaptic weight is found to *increase*, if presynaptic firing *precedes* a postsynaptic spike and decreased otherwise. In contrast to the standard *rate* models of Hebbian learning, the spike-based learning rule discussed in this paper takes these effects into account. For mathematical details and numerical simulations the reader is referred to Kempter et al. (1999).

## 2   Derivation of the Learning Equation

### 2.1   Specification of the Hebb Rule

We consider a neuron that receives input from $N \gg 1$ synapses with efficacies $J_i$, $1 \leq i \leq N$. We assume that changes of $J_i$ are induced by pre- and postsynaptic spikes. The learning rule consists of three parts. (i) Let $t_i^m$ be the time of the $m$ th input spike arriving at synapse $i$. The arrival of the spike induces the weight $J_i$ to change by an amount $w^{\mathrm{in}}$ which can be positive or negative. (ii) Let $t^n$ be the $n$ th output spike of the neuron under consideration. This event triggers the change of all $N$ efficacies by an amount $w^{\mathrm{out}}$ which can also be positive or negative. (iii) Finally, time *differences* between input spikes influence the change of the efficacies. Given a time difference $s = t_i^m - t^n$ between input and output spikes, $J_i$ is changed by an amount $W(s)$ where the *learning window* $W$ is a real valued function (Fig. 1). The learning window can be motivated by local chemical processes at the level of the synapse (Gerstner et al. 1998, Senn et al. 1999). Here we simply assume that such a learning window exist and take some (arbitrary) functional dependence $W(s)$.

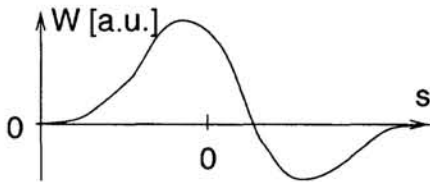

Figure 1: An example of a learning window $W$ as a function of the delay $s = t_i^m - t^n$ between a postsynaptic firing time $t^n$ and presynaptic spike arrival $t_i^m$ at synapse $i$. Note that for $s < 0$ the presynaptic spike *precedes* postsynaptic firing.

Starting at time $t$ with an efficacy $J_i(t)$, the total change $\Delta J_i(t) = J_i(t+\mathcal{T}) - J_i(t)$ in a time interval $\mathcal{T}$ is calculated by summing the contributions of all input and output spikes in the time interval $[t, t+\mathcal{T}]$. Describing the input spike train at synapse $i$ by a series of $\delta$ functions, $S_i^{\mathrm{in}}(t) = \sum_m \delta(t - t_i^m)$, and, similarly, output spikes by $S^{\mathrm{out}}(t) = \sum_n \delta(t - t^n)$, we can formulate the rules (i)–(iii):

$$\Delta J_i(t) = \int\limits_t^{t+\mathcal{T}} \mathrm{d}t' \left[ w^{\mathrm{in}} S_i^{\mathrm{in}}(t') + w^{\mathrm{out}} S^{\mathrm{out}}(t') + \int\limits_t^{t+\mathcal{T}} \mathrm{d}t'' \, W(t'' - t') S_i^{\mathrm{in}}(t'') S^{\mathrm{out}}(t') \right] \quad (1)$$

### 2.2   Separation of Time Scales

The total change $\Delta J_i(t)$ is subject to noise due to stochastic spike arrival and, possibly, stochastic generation of output spikes. We therefore study the *expected* development of the weights $J_i$, denoted by angular brackets. We make the substitution $s = t'' - t'$ on the right-hand side of (1), divide both sides by $\mathcal{T}$, and take

the expectation value:

$$
\frac{\langle \Delta J_i \rangle(t)}{\mathcal{T}} = \frac{1}{\mathcal{T}} \int_t^{t+\mathcal{T}} dt' \left[ w^{\text{in}} \langle S_i^{\text{in}} \rangle(t') + w^{\text{out}} \langle S^{\text{out}} \rangle(t') \right]
$$
$$
+ \frac{1}{\mathcal{T}} \int_t^{t+\mathcal{T}} dt' \int_{t-t'}^{t+\mathcal{T}-t'} ds \, W(s) \, \langle S_i^{\text{in}}(t'+s) \, S^{\text{out}}(t') \rangle \ . \quad (2)
$$

We may interpret $\langle S_i^{\text{in}} \rangle(t)$ for $1 \leq i \leq N$ and $\langle S^{\text{out}} \rangle(t)$ as *instantaneous* firing rates.[1] They may vary on very short time scales – shorter, e.g., than average interspike intervals. Such a model is consistent with the idea of temporal coding, since it does not rely on temporally averaged *mean* firing rates.

We note, however, that due to the integral over time on the right-hand side of (2) temporal averaging is indeed important. If $\mathcal{T}$ is much larger than typical interspike intervals, we may define *mean* firing rates $\nu_i^{\text{in}}(t) = \overline{\langle S_i^{\text{in}} \rangle}(t)$ and $\nu^{\text{out}}(t) = \overline{\langle S^{\text{out}} \rangle}(t)$ where we have used the notation $\overline{f(t)} = \mathcal{T}^{-1} \int_t^{t+\mathcal{T}} dt' \, f(t')$. The *mean* firing rates must be distinguished from the previously defined *instantaneous* rates $\langle S_i^{\text{in}} \rangle$ and $\langle S^{\text{out}} \rangle$ which are defined as an expectation value and have a high temporal resolution. In contrast, the mean firing rates $\nu_i^{\text{in}}$ and $\nu^{\text{out}}$ vary slowly (time scale of the order of $\mathcal{T}$) as a function of time.

If the learning time $\mathcal{T}$ is much larger than the width of the learning window, the integration over $s$ in (2) can be extended to run from $-\infty$ to $\infty$ without introducing a noticeable error. With the definition of a temporally averaged correlation,

$$
C_i(s;t) = \overline{\langle S_i^{\text{in}}(t+s) \, S^{\text{out}}(t) \rangle} \ , \quad (3)
$$

the last term on the right of (2) reduces to $\int_{-\infty}^{\infty} ds \, W(s) \, C_i(s;t)$. Thus, correlations between pre- and postsynaptic spikes enter spike-based Hebbian learning through $C_i$ *convolved* with the learning window $W$. We remark that the correlation $C_i(s;t)$ may change as a function of $s$ on a fast time scale. Note that, by definition, $s < 0$ implies that a presynaptic spike *precedes* the output spike – and this is when we expect (for excitatory synapses) a positive correlation between input and output.

As usual in the theory of Hebbian learning, we require learning to be a slow process. The correlation $C_i$ can then be evaluated for a constant $J_i$ and the left-hand side of (2) can be rewritten as a differential on the slow time scale of learning

$$
\frac{d}{dt} J_i(t) \equiv \dot{J}_i = w^{\text{in}} \nu_i^{\text{in}}(t) + w^{\text{out}} \nu^{\text{out}}(t) + \int_{-\infty}^{\infty} ds \, W(s) \, C_i(s;t) \ . \quad (4)
$$

## 2.3 Relation to Rate-Based Hebbian Learning

In neural network theory, the hypothesis of Hebb (Hebb 1949) is usually formulated as a learning rule where the change of a synaptic efficacy $J_i$ depends on the correlation between the mean firing rate $\nu_i^{\text{in}}$ of the $i$th presynaptic and the mean firing rate $\nu^{\text{out}}$ of a postsynaptic neuron, viz.,

$$
\dot{J}_i = a_0 + a_1 \nu_i^{\text{in}} + a_2 \nu^{\text{out}} + a_3 \nu_i^{\text{in}} \nu^{\text{out}} + a_4 (\nu_i^{\text{in}})^2 + a_5 (\nu^{\text{out}})^2 \ , \quad (5)
$$

where $a_0$, $a_1$, $a_2$, $a_3$, $a_4$, and $a_5$ are proportionality constants. Apart from the decay term $a_0$ and the 'Hebbian' term $\nu_i^{\text{in}} \nu^{\text{out}}$ proportional to the product of input and

output rates, there are also synaptic changes which are driven separately by the pre- and postsynaptic rates. The parameters $a_0, \ldots, a_5$ may depend on $J_i$. Equation (5) is a general formulation up to second order in the rates; see, e.g., (Linsker 1986).

To get (5) from (4) two approximations are necessary. First, if there are no correlations between input and output spikes apart from the correlations contained in the rates, we can approximate $\langle S_i^{in}(t + s) S^{out}(t) \rangle \approx \langle S_i^{in} \rangle (t + s) \langle S^{out} \rangle (t)$. Second, if these rates change slowly as compared to $\mathcal{T}$, then we have $C_i(s; t) \approx \nu_i^{in}(t+s) \nu^{out}(t)$. Since we have assumed that the learning time $\mathcal{T}$ is long compared to the width of the learning window, we may simplify further and set $\nu_i^{in}(t + s) \approx \nu_i^{in}(t)$, hence $\int_{-\infty}^{\infty} ds\, W(s)\, C_i(s; t) \approx \tilde{W}(0)\, \nu_i^{in}(t)\, \nu^{out}(t)$, where $\tilde{W}(0) = \int_{-\infty}^{\infty} ds\, W(s)$. We may now identify $\tilde{W}(0)$ with $a_3$. By further comparison of (5) with (4) we identify $w^{in}$ with $a_1$ and $w^{out}$ with $a_2$, and we are able to reduce (4) to (5) by setting $a_0 = a_4 = a_5 = 0$.

The above set of of assumption which is necessary to derive (5) from (4) does, however, not hold in general. According to the results of Markram et al. (1997) the width of the learning window in cortical pyramidal cells is in the range of $\approx 100\,$ms. A *mean* rate formulation thus requires that all changes of the activity are slow on a time scale of $100\,$ms. This is not necessarily the case. The existence of oscillatory activity in the cortex in the range of $50\,$Hz implies activity changes every $20\,$ms. Much faster activity changes on a time scale of $1\,$ms and below are found in the auditory system (e.g., Carr 1993). Furthermore, beyond the correlations between mean activities additional correlations between spikes may exist; see below. Because of all these reasons, the learning rule (5) in the simple rate formulation is insufficient. In the following we will study the full spike-based learning equation (4).

## 3   Stochastically Spiking Neurons

### 3.1   Poisson Input and Stochastic Neuron Model

To proceed with the analysis of (4) we need to determine the correlations $C_i$ between input spikes at synapse $i$ and output spikes. The correlations depend strongly on the neuron model under consideration. To highlight the main points of learning we study a linear inhomogeneous Poisson neuron as a toy model. Input spike trains arriving at the $N$ synapses are statistically independent and generated by an inhomogeneous Poisson process with time-dependent intensities $\langle \dot{S}_i^{in} \rangle (t) = \lambda_i^{in}(t)$, with $1 \leq i \leq N$. A spike arriving at $t_i^m$ at synapse $i$, evokes a postsynaptic potential (PSP) with time course $\epsilon(t - t_i^m)$ which we assume to be excitatory (EPSP). The amplitude is given by the synaptic efficacy $J_i(t) > 0$. The membrane potential $u$ of the neuron is the linear superposition of all contributions

$$u(t) = u_0 + \sum_{i=1}^{N} \sum_m J_i(t)\, \epsilon(t - t_i^m) \qquad (6)$$

where $u_0$ is the resting potential. Output spikes are assumed to be generated stochastically with a time dependent rate $\lambda^{out}(t)$ which depends linearly upon the membrane potential

$$\lambda^{out}(t) = \beta\,[u(t)]_+ = \nu_0 + \sum_{i=1}^{N} \sum_m J_i(t)\, \epsilon(t - t_i^m). \qquad (7)$$

with a linear function $\beta[u]_+ = \beta_0 + \beta_1\, u$ for $u > 0$ and zero otherwise. After the second equality sign, we have formally set $\nu_0 = u_0 + \beta_0$ and $\beta_1 = 1$. $\nu_0 >$ can

be interpreted as the spontaneous firing rate. For excitatory synapses a negative $u$ is impossible and that's what we have used after the second equality sign. The sums run over all spike arrival times at all synapses. Note that the spike generation process is independent of previous output spikes. In particular, the Poisson model does not include refractoriness.

In the context of (4), we are interested in the expectation values for input and output. The expected input is $\langle S_i^{\text{in}} \rangle(t) = \lambda_i^{\text{in}}(t)$. The expected output is

$$\langle S^{\text{out}} \rangle(t) = \nu_0 + \sum_i J_i(t) \int_0^\infty ds\, \epsilon(s)\, \lambda_i^{\text{in}}(t - s) \ , \tag{8}$$

The expected output rate in (8) depends on the convolution of $\epsilon$ with the input rates. In the following we will denote the convolved rates by $\Lambda_i^{\text{in}}(t) = \int_0^\infty ds\, \epsilon(s) \lambda_i^{\text{in}}(t-s)$.

Next we consider the expected correlations between input and output, $\langle S_i^{\text{in}}(t + s)\, S^{\text{out}}(t) \rangle$, which we need in (3):

$$\langle S_i^{\text{in}}(t + s)\, S^{\text{out}}(t) \rangle = \lambda_i^{\text{in}}(t + s)\, [\nu_0 + J_i(t)\, \epsilon(-s) + \sum_j J_j(t)\, \Lambda_j^{\text{in}}(t)] \ . \tag{9}$$

The first term inside the square brackets is the spontaneous output rate. The second term is the specific contribution of an input spike at time $t + s$ to the output rate at $t$. It vanishes for $s > 0$ (Fig. 2). The sum in (9) contains the mean contributions of all synapses to an output spike at time $t$. Inserting (9) in (3) and assuming the weights $J_j$ to be constant in the time interval $[t, t + \mathcal{T}]$ we obtain

$$C_i(s; t) = \sum_j J_j(t)\, \overline{\lambda_i^{\text{in}}(t + s)\, \Lambda_j^{\text{in}}(t)} + \overline{\lambda_i^{\text{in}}(t + s)\, [\nu_0 + J_i(t)\, \epsilon(-s)]} \ . \tag{10}$$

For excitatory synapses, the second term gives for $s < 0$ a positive contribution to the correlation function – as it should be. (Recall that $s < 0$ means that a presynaptic spike precedes postsynaptic firing.)

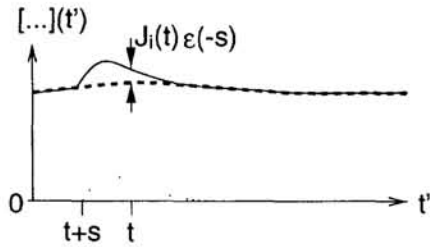

Figure 2: Interpretation of the term in square brackets in (9). The dotted line is the contribution of an input spike at time $t + s$ to the output rate as a function of $t'$, viz., $J_i(t)\, \epsilon(t' - t - s)$. Adding this to the mean rate contribution, $\nu_0 + \sum_j J_j(t')\, \Lambda_j^{\text{in}}(t')$ (dashed line), we obtain the rate inside the square brackets of (9) (full line). At time $t' = t$ the contribution of an input spike at time $t + s$ is $J_i(t)\, \epsilon(-s)$.

## 3.2 Learning Equation

The assumption of identical and constant mean input rates, $\overline{\lambda_i^{\text{in}}(t)} = \nu_i^{\text{in}}(t) = \nu^{\text{in}}$ for all $i$, reduces the number of free parameters in (4) and eliminates all effects of rate coding. We introduce $\Gamma_i^{\text{in}}(t) := [\tilde{W}(0)]^{-1} \int_{-\infty}^\infty ds\, W(s)\lambda_i^{\text{in}}(t + s)$ and define

$$Q_{ij}(t) = \tilde{W}(0)\, \overline{[\Gamma_i^{\text{in}}(t) - \nu^{\text{in}}]\, [\Lambda_j^{\text{in}}(t) - \nu^{\text{in}}]} \ . \tag{11}$$

Using (8), (10), (11) in (4) we find for the evolution on the slow time scale of learning

$$\dot{J}_i(t) = k_1 + \sum_j J_j(t)\, [Q_{ij}(t) + k_2 + k_3\, \delta_{ij}] \ , \quad \text{where} \tag{12}$$

$$k_1 = [w^{\text{out}} + \tilde{W}(0)\,\nu^{\text{in}}]\,\nu_0 + w^{\text{in}}\,\nu^{\text{in}} \tag{13}$$

$$k_2 = [w^{\text{out}} + \tilde{W}(0)\,\nu^{\text{in}}]\,\nu^{\text{in}} \tag{14}$$

$$k_3 = \nu^{\text{in}} \int \mathrm{d}s\,\epsilon(-s)\,W(s)\,. \tag{15}$$

## 4  Discussion

Equation (12), which is the central result of our analysis, describes the expected dynamics of synaptic weights for a spike-based Hebbian learning rule (1) under the assumption of a linear inhomogeneous Poisson neuron. Linsker (1986) has derived a mathematically equivalent equation starting from (5) and a linear graded response neuron, a rate-based model. An equation of this type has been analyzed by MacKay and Miller (1990). The difference between Linsker's equation and (12) is, apart from a slightly different notation, the term $k_3\,\delta_{ij}$ and the interpretation of $Q_{ij}$.

### 4.1  Interpretation of $Q_{ij}$

In (12) correlations between spikes on time scales down to milliseconds or below can enter the driving term $Q_{ij}$ for structure formation; cf. (11). In contrast to that, Linsker's ansatz is based on a firing rate description, where the term $Q_{ij}$ contains correlations between *mean firing rates* only. In his $Q_{ij}$ term, mean firing rates take the place of $\Gamma_i^{\text{in}}$ and $\Lambda_i^{\text{in}}$. If we use a standard interpretation of rate coding, a mean firing rate corresponds to a temporally averaged quantity with an averaging window or a hundred milliseconds or more.

Formally, we could define mean rates by temporal averaging with either $\epsilon(s)$ or $W(s)$ as the averaging window. In this sense, Linsker's 'rates' have been made more precise by (11). Note, however, that (11) is asymmetric: one of the rates should be convolved with $\epsilon$, the other one with $W$.

### 4.2  Relevance of the $k_3$ term

The most important difference between Linsker's rate-based learning rule and our Eq. (12) is the existence of a term $k_3 \neq 0$. We now argue that for a *causal* chain of events $k_3 \propto \int \mathrm{d}x\,\epsilon(x)\,W(-x)$ must be positive. [We have set $x = -s$ in (15).] First, without loss of generality, the integral can be restricted to $x > 0$ since $\epsilon(x)$ is a response kernel and vanishes for $x < 0$. For excitatory synapses, $\epsilon(x)$ is positive for $x > 0$. Second, experiments on excitatory synapses show that $W(s)$ is positive for $s < 0$ (Markram et al. 1997, Zhang et al. 1998). Thus the integral $\int \mathrm{d}x\,\epsilon(x)\,W(-x)$ is positive – and so is $k_3$.

There is also a more general argument for $k_3 > 0$ based on a literal interpretation of Hebb's statement (Hebb 1949). Let us recall that $s < 0$ in (15) means that a presynaptic spike *precedes* postsynaptic spiking. For excitatory synapses, a presynaptic spike which precedes postsynaptic firing may be the *cause* of the postsynaptic activity. [As Hebb puts it, it has 'contributed in firing the postsynaptic cell'.] Thus, the Hebb rule 'predicts' that for excitatory synapses $W(s)$ is positive for $s < 0$. Hence, $k_3 = \nu^{\text{in}} \int \mathrm{d}s\,\epsilon(-s)\,W(s) > 0$ as claimed above.

A positive $k_3$ term in (12) gives rise to an exponential growth of weights. Thus any existing structure in the distribution of weights is enhanced. This contributes to the *stability* of weight distributions, especially when there are few and strong synapses (Gerstner et al. 1996).

## 4.3 Intrinsic Normalization

Let us suppose that no input synapse is special and impose the (weak) condition that $N^{-1} \sum_i Q_{ij} = Q_0 > 0$ independent of the synapse index $j$. We find then from (12) that the average weight $J_0 := N^{-1} \sum_i J_i$ has a fixed point $J_0 = -k_1/[Q_0 + k_2 + N^{-1}k_3]$. The fixed point is stable if $Q_0 + k_2 + N^{-1}k_3 < 0$. We have shown above that $k_3 > 0$. Furthermore, $Q_0 > 0$ according to our assumption. The only way to enforce stability is therefore a term $k_2$ which is sufficiently negative. Let us now turn to the definition of $k_2$ in (14). To achieve $k_2 < 0$, either $\tilde{W}(0)$ (the integral over $W$) must be sufficiently negative; this corresponds to a learning rule which is, on the average, anti-Hebbian. Or, for $\tilde{W}(0) > 0$, the linear term $w^{out}$ in (1) must be sufficiently negative. In addition, for excitatory synapses a reasonable fixed point $J_0$ has to be positive. For a stable fixed point this is only possible for $k_1 > 0$, which, in turn, implies $w^{in}$ to be sufficiently positive; cf. (13).

Intrinsic normalization of synaptic weights is an interesting property, since it allows neurons to stay at an optimal operating point even while synapses are changing. Auditory neurons may use such a mechanism to stay during learning in the regime where coincidence detection is possible (Gerstner et al. 1996, Kempter et al. 1998). Cortical neurons might use the same principles to operate in the regime of high variability (Abbott, invited NIPS talk, this volume).

## 4.4 Conclusions

Spike-based learning is different from simple rate-based learning rules. A spike-based learning rule can pick up correlations in the input on a millisecond time scale. Mathematically, the main difference to rate-based Hebbian learning is the existence of a $k_3$ term which accounts for the causal relation between input and output spikes. Correlations between input and output spikes on a millisecond time scale play a role and tend to stabilize existing strong synapses.

## Footnotes

\*email: kempter@physik.tu-muenchen.de (corresponding author)

[1]An example of rapidly changing instantaneous rates can be found in the auditory system. The auditory nerve carries noisy spike trains with a stochastic intensity modulated at the frequency of the applied acoustic tone. In the barn owl, a significant modulation of the rates is seen up to a frequency of 8 kHz (e.g., Carr 1993).

# References

Abeles M., 1994, In Domany E. et al., editors, *Models of Neural Networks II*, pp. 121–140, New York. Springer.

Bialek W. et al., 1991, *Science*, 252:1855–1857.

Carr C. E., 1993, *Annu. Rev. Neurosci.*, 16:223–243.

Gerstner W. et al., 1996, *Nature*, 383:76–78.

Gerstner W. et al., 1998, In W. Maass and C. M. Bishop., editors, *Pulsed Neural Networks*, pp. 353-377, Cambridge. MIT-Press.

Hebb D. O., 1949, *The Organization of Behavior*. Wiley, New York.

Kempter R. et al., 1998, *Neural Comput.*, 10:1987–2017.

Kempter R. et al., 1999, *Phys. Rev. E*, In Press.

Linsker R., 1986, *Proc. Natl. Acad. Sci. USA*, 83:7508–7512.

MacKay D. J. C., Miller K. D., 1990, *Network*, 1:257–297.

Markram H. et al., 1997, *Science*, 275:213–215.

Senn W. et al., 1999, *preprint*, Univ. Bern.

Wimbauer S. et al., 1997, *Biol. Cybern.*, 77:453–461.

Zhang L.I. et al., 1998, *Nature*, 395:37–44
